# A Bayesian Model Predicts Human Parse Preference and Reading Times in Sentence Processing

**Srini Narayanan**
**SRI International and ICSI Berkeley**
snarayan@cs.berkeley.edu

**Daniel Jurafsky**
**University of Colorado, Boulder**
jurafsky@colorado.edu

## Abstract

Narayanan and Jurafsky (1998) proposed that human language comprehension can be modeled by treating human comprehenders as Bayesian reasoners, and modeling the comprehension process with Bayesian decision trees. In this paper we extend the Narayanan and Jurafsky model to make further predictions about reading time given the probability of difference parses or interpretations, and test the model against reading time data from a psycholinguistic experiment.

## 1 Introduction

Narayanan and Jurafsky (1998) proposed that human language comprehension can be modeled by treating human comprehenders as Bayesian reasoners, and modeling the comprehension process with Bayesian decision trees. In this paper, we show that the model accounts for parse-preference and reading time data from a psycholinguistic experiment on reading time in ambiguous sentences.

Parsing, (generally called 'sentence processing' when we are referring to human parsing), is the process of building up syntactic interpretations for a sentence from an input sequence of written or spoken words. Ambiguity is extremely common in parsing problems, and previous research on human parsing has focused on showing that many factors play a role in choosing among the possible interpretations of an ambiguous sentence.

We will focus in this paper on a syntactic ambiguity phenomenon which has been repeatedly investigated: the main-verb (MV), reduced relative (RR) local ambiguity (Frazier & Rayner, 1987; MacDonald, Pearlmutter, & Seidenberg, 1994; McRae, Spivey-Knowlton, & Tanenhaus, 1998, inter alia) In this ambiguity, a prefix beginning with a noun-phrase and an ambiguous verb-form could either be continued as a main clause (as in 1a), or turn out to be a relative clause modifier of the first noun phrase (as in 1b).

1. a. The cop arrested the forger.
   b. The cop arrested by the detective was guilty of taking bribes.

Many factors are known to influence human parse preferences. One such factor is the different lexical/morphological frequencies of the simple past and participial forms of the ambiguous verbform (*arrested*, in this case). Trueswell (1996) found that verbs like *searched*, with a frequency-based preference for the simple past form, caused readers to prefer the

main clause interpretation, while verbs like *selected*, had a preference for a participle reading, and supported the reduced relative interpretation.

The transitivity preference of the verb also plays a role in human syntactic disambiguation. Some verbs are preferably transitive, where others are preferably intransitive. The reduced relative interpretation, since it involves a passive structure, requires that the verb be transitive. MacDonald, Pearlmutter, and Seidenberg (1994), Trueswell, Tanenhaus, and Kello (1994) and other have shown that verbs which are biased toward an intransitive interpretation also bias readers toward a main clause interpretation.

Previous work has shown that a competition-integration model developed by Spivey-Knowlton (1996) could model human parse preference in reading ambiguous sentences (McRae *et al.*, 1998). While this model does a nice job of accounting for the reading-time data, it and similar 'constraint-based' models rely on a complex set of feature values and factor weights which must be set by hand. Narayanan and Jurafsky (1998) proposed an alternative Bayesian approach for this constraint-combination problem. A Bayesian approach offers a well-understood formalism for defining probabilistic weights, as well as for combining those weights. Their Bayesian model is based on the probabilistic beam-search of Jurafsky (1996), in which each interpretation receives a probability, and interpretations were pruned if they were much worse than the best interpretation. The model predicted large increases in reading time when unexpected words appeared which were only compatible with a previously-pruned interpretation. The model was thus only able to characterize very gross timing effects caused by pruning of interpretations.

In this paper we extend this model's predictions about reading time to other cases where the best interpretation turns out to be incompatible with incoming words. In particular, we suggest that any evidence which causes the probability of the best interpretation to drop below its next competitor will also cause increases in reading time.

## 2   The Experimental Data

We test our model on the reading time data from McRae *et al.* (1998), an experiment focusing on the effect of thematic fit on syntactic ambiguity resolution. The thematic role of noun phrase "the cop" in the prefix "The cop arrested" is ambiguous. In the continuation "The cop arrested the crook", the cop is the *agent*. In the continuation "The cop arrested by the FBI agent was convicted for smuggling drugs", the cop is the *theme*. The probabilistic relationship between the noun and the head verb ("arrested") biases the thematic disambiguation decision. For example, "cop" is a more likely *agent* for "arrest", while "crook" is a more likely *theme*. McRae *et al.* (1998) showed that this 'thematic fit' between the noun and verb affected phrase-by-phrase reading times in sentences like the following:

2.  a.  The cop / arrested by / the detective / was guilty / of taking / bribes.
    b.  The crook / arrested by / the detective / was guilty / of taking / bribes.

In a series of experiment on 40 verbs, they found that sentences with good agents (like *cop* in 2a) caused longer reading times for the phrase *the detective* than sentences with good themes (like *crook* in 2b). Figure 1 shows that at the initial noun phrase, reading time is lower for good-agent sentences than good-patient sentences. But at the NP after the word "by", reading time is lower for good-patient sentences than good-agent sentences.[1]

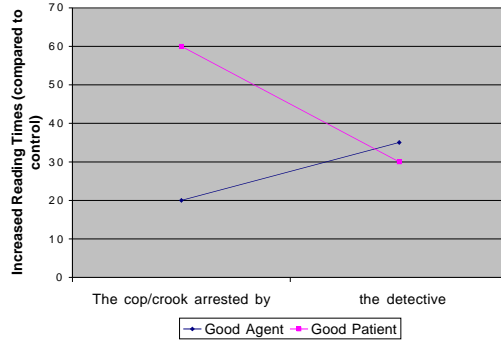

Figure 1: Self-paced reading times (from Figure 6 of McRae *et al.* (1998))

After introducing our model in the next section, we show that it predicts this cross-over in reading time; longer reading time for the initial NP in good-patient sentences, but shorter reading time for the post-"by" NP in good-patient sentences.

## 3   The Model and the Input Probabilities

In the Narayanan and Jurafsky (1998) model of sentence processing, each interpretation of an ambiguous sentence is maintained in parallel, and associated with a probability which can be computed via a Bayesian belief net. The model pruned low-probability parses, and hence predicted increases in reading time when reading a word which did not fit into any available parse. The current paper extends the Narayanan and Jurafsky (1998) model's predictions about reading time. The model now also predicts extended reading time whenever an input word causes the best interpretation to drop in probability enough to switch in rank with another interpretation.

The model consists of a set of probabilities expressing constraints on sentence processing, and a network that represents their independence relations:

| Data | Source |
|------|--------|
| P(Agent\|verb, initial NP) | McRae *et al.* (1998) |
| P(Patient\|verb, initial NP) | McRae *et al.* (1998) |
| P(Participle\|verb) | British National Corpus counts |
| P(SimplePast\|verb) | British National Corpus counts |
| P(transitive\|verb) | TASA corpus counts |
| P(intransitive\|verb) | TASA corpus counts |
| P(RR\|initial NP, verb-ed, by) | McRae *et al.* (1998) (.8, .2) |
| P(RR\|initial NP, verb-ed, by,the) | McRae *et al.* (1998) (.875. .125) |
| P(Agent\|initial NP, verb-ed, by, the, NP) | McRae *et al.* (1998) (4.6 average) |
| P(MC\| SCFG prefix) | SCFG counts from Penn Treebank |
| P(RR\| SCFG prefix) | SCFG counts from Penn Treebank |

The first constraint expresses the probability that the word "cop", for example, is an agent, given that the verb is "arrested". The second constraint expresses the probability that it is a patient. The third and fourth constraints express the probability that the "-ed" form of the verb is a participle versus a simple past form (for example P(Participle | "arrest")=.81). These were computed from the POS-tagged British National Corpus. Verb transitivity probabilities were computed by hand-labeling subcategorization of 100 examples of each verb in the TASA corpus. (for example P(transitive | "entertain")=.86). Main clause prior probabilities were computed by using an SCFG with rule probabilities trained on the Penn

Treebank version of the Brown corpus. See Narayanan and Jurafsky (1998) and Jurafsky (1996) for more details on probability computation.

# 4    Construction Processing via Bayes nets

Using Belief nets to model human sentence processing allows us to a) quantitatively evaluate the impact of different independence assumptions in a uniform framework, b) directly model the impact of highly structured linguistic knowledge sources with local conditional probability tables, while well known algorithms for updating the Belief net (Jensen (1995)) can compute the global impact of new evidence, and c) develop an on-line interpretation algorithm, where partial input corresponds to partial evidence on the network, and the update algorithm appropriately marginalizes over unobserved nodes. So as evidence comes in incrementally, different nodes are instantiated and the posterior probability of different interpretations changes appropriately.

The crucial insight of our Belief net model is to view specific interpretations as *values* of *latent variables* that render top-down ( $e^+$ ) and bottom-up evidence ( $e^-$ ) conditionally independent (d-separate them (Pearl, 1988)). Thus syntactic, lexical, argument structure, and other contextual information acts as *prior* or *causal* support for an interpretation, while bottom-up phonological or graphological and other perceptual information acts as *likelihood*, *evidential*, or *diagnostic* support.

To apply our model to on-line disambiguation, we assume that there are a set of interpretations ( $(i_1, \ldots i_n) \in I$ ) that are consistent with the input data. At different stages of the input, we compute the posterior probabilities of the different interpretations given the top down and bottom-up evidence seen so far. [2]

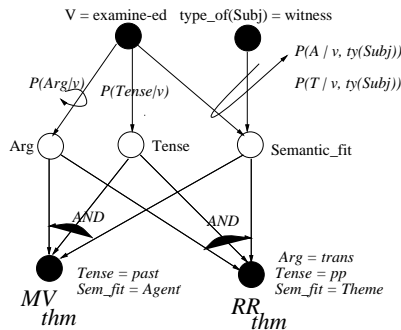

Figure 2: The Belief net that represents lexical and thematic support for the two interpretations.

Figure 2 reintroduces the basic structure of our belief net model from Narayanan and Jurafsky (1998). Our model requires conditional probability distributions specifying the preference of every verb for different argument structures, as well its preference for different tenses. We also compute the semantic fit between possible fillers in the input and different conceptual roles of a given predicate. As shown in Figure 2, the $MV$ and $RR$ interpretations require the conjunction of specific values corresponding to tense, semantic fit and argument structure features. Note that only the $RR$ interpretation requires the transitive argument structure.

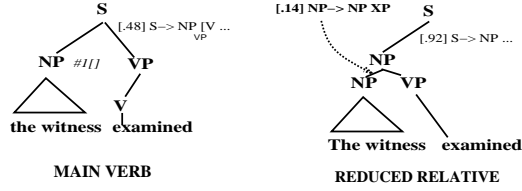

MAIN VERB&emsp;&emsp;&emsp;&emsp;&emsp;REDUCED RELATIVE

Figure 3: The partial syntactic parse trees for the $MV$ and the $RR$ interpretations assuming an $SCFG$ generating grammar.

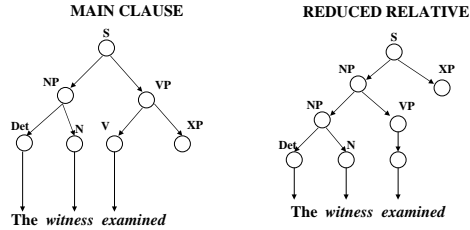

Figure 4: The Bayes nets for the partial syntactic parse trees

The conditional probability of a construction given top-down syntactic evidence $P(c|e)$ is relatively simple to compute in an augmented-stochastic-context-free formalism (partial parse trees shown in Figure 3 and the corresponding bayes net in Figure 4). Recall that the $SCFG$ prior probability gives the conditional probability of the right hand side of a rule given the left hand side. The Inside/Outside algorithm applied to a fixed parse tree structure is obtained exactly by casting parsing as a special instance of belief propagation. The correspondences are straightforward a) the parse tree is interpreted as a belief network. b) the non-terminal nodes correspond to random variables, the range of the variables being the non-terminal alphabet, c) the grammar rules define the conditional probabilities linking parent and child nodes, d) the $S$ nonterminal at the root, as well as the terminals at the leaves represent conditioning evidence to the network, and e) Conditioning on this evidence produces exactly the conditional probabilities for each nonterminal node in the parse tree and the joint probability distribution of the parse.[3]

The overall posterior ratio requires propagating the *conjunctive* impact of syntactic and lexical/thematic sources on our model. Furthermore, in computing the conjunctive impact of the lexical/thematic and syntactic support to compute $MV$ and $RR$, we use the NOISY-AND (assumes exception independence) model (Pearl, 1988) for combining conjunctive sources. In the case of the $RR$ and $MV$ interpretations. At various points, we compute the posterior support for the different interpretations using the following equation. $P(X) = P(X|Syn, Lex, Thm) = P(X|Syn) \times P(X|lex, thm)$. The first term is

the syntactic support while the second is the lexical and thematic support for a particular interpretation ($X \in RR, MV$).

## 5  Model results

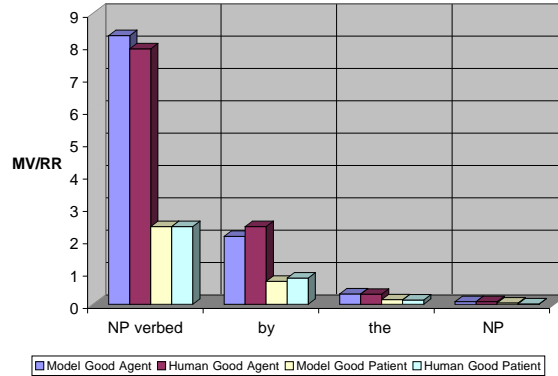

Figure 5: Completion data

We tested our model on sentences with the different verbs in McRae *et al.* (1998). For each verb, we ran our model on sentences with Good Agents (GA) and Good Patients (GP) for the initial NP. Our model results are consistent with the on-line disambiguation studies with human subjects (human performance data from McRae *et al.* (1998)) and show that a Bayesian implementation of probabilistic evidence combination accounts for garden-path disambiguation effects.

Figure 5 shows the first result that pertains to the model predictions of how thematic fit might influence sentence completion times. Our model shows close correspondence to the human judgements about whether a specific ambiguous verb was used in the Main Clause (MV) or reduced relative (RR) constructions. The human and model predictions were conducted at the *verb* (The crook arrested), *by* (the crook arrested by), *the* (the crook arrested by the) and Agent NP (the crook arrested by the detective). As in McRae *et al.* (1998) the data shows that thematic fit clearly influenced the gated sentence completion task. The probabilistic account further captured the fact that at the *by* phrase, the posterior probability of producing an RR interpretation increased sharply, thematic fit and other factors

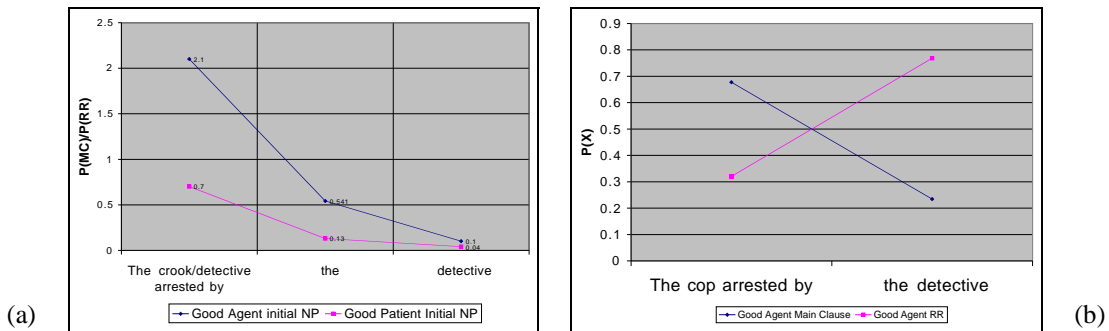

(a)          (b)

Figure 6: a) MV/RR for the ambiguous region showing a flip for the Good Agent (ga) case. b) P(MV) and P(RR) for the Good Patient and Good Agent cases.

influenced both the sharpness and the magnitude of the increase.

The second result pertains to on-line reading times. Figure 6 shows how the human reading time reduction effects (reduced compared to unreduced interpretations) increase for Good Agents (GA) but decrease for Good Patients in the ambiguous region. This explains the reading time data in Figure 1. Our model predicts this larger effect from the fact that the most probable interpretation for the Good Agent case *flips* from the MV to the RR interpretation in this region. No such flip occurs for the Good Patient (GP) case. In Figure 6(a), we see that the GP results already have the MV/RR ratio less than one (the RR interpretation is superior) while a flip occurs for the GA sentences (from the initial state where MV/RR $> 1$ to the final state where MV/RR $< 1$). Figure 6 (b) shows a more detailed view of the GA sentences showing the crossing point where the *flip* occurs. This finding is fairly robust ($73\%$ of GA examples) and directly predicts reading time difficulties.

# 6    Conclusion

We have shown that a Bayesian model of human sentence processing is capable of modeling reading time data from a syntactic disambiguation task. A Bayesian model extends current constraint-satisfaction models of sentence processing with a principled way to weight and combine evidence. Bayesian models have not been widely applied in psycholinguistics. To our knowledge, this is the first study showing a direct correspondence between the time course of maintaining the best a posteriori interpretation and reading time difficulty.

We are currently exploring how our results on *flipping* of preferred interpretation could be combined with Hale (2001)'s proposal that reading time correlates with *surprise* (a surprising (low probability) word leads to large amounts of probability mass to be pruned) to arrive at a structured probabilistic account of a wide variety of psycholinguistic data.

## Footnotes

[1] In order to control for other influences on timing, McRae *et al.* (1998) actually report reading time deltas between a reduced relative and non-reduced relative for. It is these deltas, rather than raw reading times, that our model attempts to predict.

[2]In this paper, we will focus on the support from thematic, and syntactic features for the Reduced Relative (RR) and Main Verb (MV) interpretations at different stages of the input for the examples we saw earlier. So we will have two interpretations $i_1, i_2 \in I$ where $P(i_1|e^+, e^-) = MV, P(i_2|e^+, e^-) = RR$.

[3]One complication is that the the conditional distribution in a parse tree $P(Y, Z|X)$ is not the product distribution $P(Y|X)P(Z|X)$ (it is the conjunctive distribution). However, it is possible to generalize the belief propagation equations to admit conjunctive distributions $P(Y, Z|X)$ and $P(X, V|U)$. The diagnostic (inside) support becomes $\lambda(x) = \sum_{y,z} \lambda(y)\lambda(z)P(y, z|x)$ and the causal support becomes $\pi(x) = \beta \sum_{u,v} \pi(u)\lambda(v)P(x, v|u)$ (details can be found at http://www.icsi.berkeley.edu/ snarayan/scfg.ps).

# References

Frazier, L., & Rayner, K. (1987). Resolution of syntactic category ambiguities: Eye movements in parsing lexically ambiguous sentences. *Journal of Memory and Language*, *26*, 505–526.

Hale, J. (2001). A probabilistic earley parser as a psycholinguistic model. *Proceedings of NAACL-2001*.

Jensen, F. (1995). *Bayesian Networks*. Springer-Verlag.

Jurafsky, D. (1996). A probabilistic model of lexical and syntactic access and disambiguation. *Cognitive Science*, *20*, 137–194.

MacDonald, M.C., Pearlmutter, N.J., & Seidenberg, M.(1994). The lexical nature of syntactic ambiguity resolution. *Psychological Review*, *101*, 676-703.

McRae, K., Spivey-Knowlton, M., & Tanenhaus, M. K.(1998). Modeling the effect of thematic fi t (and other constraints) in on-line sentence comprehension. *Journal of Memory and Language*,*38*, 283–312.

Narayanan, S., & Jurafsky, D. (1998). Bayesian models of human sentence processing. In *COGSCI-98*, pp. 752–757 Madison, WI. Lawrence Erlbaum.

Pearl, J. (1988). *Probabilistic Reasoning in Intelligent Systems: Networks of Plausible Inference*. Morgan Kaufman, San Mateo, Ca.

Spivey-Knowlton, M.(1996). Integration of visual and linguistic information: Human data and model simulations. *Ph.D. Thesis*, *University of Rochester*, 1996.

Trueswell, J. C.(1996). The role of lexical frequency in syntactic ambiguity resolution. *Journal of Memory and Language*, *35*, 566-585.

Trueswell, J. C., Tanenhaus, M. K., & Kello, C. (1994). Verb-specific constraints in sentence processing: Separating effects of lexical preference from garden-paths. *Journal of Experimental Pyschology: Learning, Memory and Cognition*, *19(3)*, 528–553.
